# Periodic Component Analysis: An Eigenvalue Method for Representing Periodic Structure in Speech

**Lawrence K. Saul and Jont B. Allen**
{lsaul, jba}@research.att.com
AT&T Labs, 180 Park Ave, Florham Park, NJ 07932

## Abstract

An eigenvalue method is developed for analyzing periodic structure in speech. Signals are analyzed by a matrix diagonalization reminiscent of methods for principal component analysis (PCA) and independent component analysis (ICA). Our method—called periodic component analysis ($\pi$CA)—uses constructive interference to enhance periodic components of the frequency spectrum and destructive interference to cancel noise. The front end emulates important aspects of auditory processing, such as cochlear filtering, nonlinear compression, and insensitivity to phase, with the aim of approaching the robustness of human listeners. The method avoids the inefficiencies of autocorrelation at the pitch period: it does not require long delay lines, and it correlates signals at a clock rate on the order of the actual pitch, as opposed to the original sampling rate. We derive its cost function and present some experimental results.

## 1  Introduction

Periodic structure in the time waveform conveys important cues for recognizing and understanding speech[1]. At the end of an English sentence, for example, rising versus falling pitch indicates the asking of a question; in tonal languages, such as Chinese, it carries linguistic information. In fact, early in the speech chain—prior to the recognition of words or the assignment of meaning—the auditory system divides the frequency spectrum into periodic and non-periodic components. This division is geared to the recognition of phonetic features[2]. Thus, a voiced fricative might be identified by the presence of periodicity in the lower part of the spectrum, but not the upper part. In complicated auditory scenes, periodic components of the spectrum are further segregated by their fundamental frequency[3]. This enables listeners to separate simultaneous speakers and explains the relative ease of separating male versus female speakers, as opposed to two recordings of the same voice[4].

The pitch and voicing of speech signals have been extensively studied[5]. The simplest method to analyze periodicity is to compute the autocorrelation function on sliding windows of the speech waveform. The peaks in the autocorrelation function provide estimates of the pitch and the degree of voicing. In clean wideband speech, the pitch of a speaker can be tracked by combining a peak-picking procedure on the autocorrelation function with some form of smoothing[6], such as dynamic programming. This method, however,

does not approach the robustness of human listeners in noise, and at best, it provides an extremely gross picture of the periodic structure in speech. It cannot serve as a basis for attacking harder problems in computational auditory scene analysis, such as speaker separation[7], which require decomposing the frequency spectrum into its periodic and non-periodic components.

The correlogram is a more powerful method for analyzing periodic structure in speech. It looks for periodicity in narrow frequency bands. Slaney and Lyon[8] proposed a perceptual pitch detector that autocorrelates multichannel output from a model of the auditory periphery. The auditory model includes a cochlear filterbank and periodicity-enhancing nonlinearities. The information in the correlogram is summed over channels to produce an estimate of the pitch. This method has two compelling features: (i) by measuring autocorrelation, it produces pitch estimates that are insensitive to phase changes across channels; (ii) by working in narrow frequency bands, it produces estimates that are robust to noise. This method, however, also has its drawbacks. Computing multiple autocorrelation functions is expensive. To avoid aliasing in upper frequency bands, signals must be correlated at clock rates much higher than the actual pitch. From a theoretical point of view, it is unsatisfying that the combination of information across channels is not derived from some principle of optimality. Finally, in the absence of conclusive evidence for long delay lines (~10 ms) in the peripheral auditory system, it seems worthwhile—for both scientists and engineers—to study ways of detecting periodicity that do not depend on autocorrelation.

In this paper, we develop an eigenvalue method for analyzing periodic structure in speech. Our method emulates important aspects of auditory processing but avoids the inefficiencies of autocorrelation at the pitch period. At the same time, it is highly robust to narrowband noise and insensitive to phase changes across channels. Note that while certain aspects of the method are biologically inspired, its details are not intended to be biologically realistic.

## 2   Method

We develop the method in four stages. These stages are designed to convey the main technical ideas of the paper: (i) an eigenvalue method for combining and enhancing weakly periodic signals; (ii) the use of Hilbert transforms to compensate for phase changes across channels; (iii) the measurement of periodicity by efficient sinusoidal fits; and (iv) the hierarchical analysis of information across different frequency bands.

### 2.1   Cross-correlation of critical bands

Consider the multichannel output of a cochlear filterbank. If the input to this filterbank consists of noisy voiced speech, the output will consist of weakly periodic signals from different critical bands. Can we combine these signals to enhance the periodic signature of the speaker's pitch? We begin by studying a mathematical idealization of the problem. Given $n$ real-valued signals, $\{x_i(t)\}_{i=1}^n$, what linear combination $s(t) = \sum_i w_i x_i(t)$ maximizes the periodic structure at some fundamental frequency $f_0$, or equivalently, at some pitch period $\tau = 1/f_0$? Ideally, the linear combination should use constructive interference to enhance periodic components of the spectrum and destructive interference to cancel noise. We measure the periodicity of the combined signal by the cost function:

$$\varepsilon(w, \tau) = \frac{\sum_t |s(t+\tau) - s(t)|^2}{\sum_t |s(t)|^2} \quad \text{with} \quad s(t) = \sum_i w_i x_i(t). \tag{1}$$

Here, for simplicity, we have assumed that the signals are discretely sampled and that the period $\tau$ is an integer multiple of the sampling interval. The cost function $\varepsilon(w, \tau)$ measures the normalized prediction error, with the period $\tau$ acting as a prediction lag. Expanding the

right hand side in terms of the weights $w_i$ gives:

$$\varepsilon(\boldsymbol{w}, \tau) = \frac{\sum_{ij} w_i w_j A_{ij}(\tau)}{\sum_{ij} w_i w_j B_{ij}}, \tag{2}$$

where the matrix elements $A_{ij}(\tau)$ are determined by the cross-correlations,

$$A_{ij}(\tau) = \sum_t \Big[ x_i(t)x_j(t) + x_i(t+\tau)x_j(t+\tau) - x_i(t)x_j(t+\tau) - x_i(t+\tau)x_j(t) \Big],$$

and the matrix elements $B_{ij}$ are the equal-time cross-correlations, $B_{ij} = \sum_t x_i(t)x_j(t)$. Note that the denominator and numerator of eq. (2) are both quadratic forms in the weights $w_i$. By the Rayleigh-Ritz theorem of linear algebra, the weights $w_i$ minimizing eq. (2) are given by the eigenvector of the matrix $B^{-1}A(\tau)$ with the smallest eigenvalue. For fixed $\tau$, this solution corresponds to the global minimum of the cost function $\varepsilon(\boldsymbol{w}, \tau)$. Thus, matrix diagonalization (or simply computing the bottom eigenvector, which is often cheaper) provides a definitive answer to the above problem.

The matrix diagonalization which optimizes eq. (2) is reminiscent of methods for principal component analysis (PCA) and independent component analysis (ICA)[9]. Our method—which by analogy we call periodic component analysis ($\pi$CA)—uses an eigenvalue principle to combine periodicity cues from different parts of the frequency spectrum.

## 2.2   Insensitivity to phase

The eigenvalue method in the previous section has one obvious shortcoming: it cannot compensate for phase changes across channels. In particular, the real-valued linear combination $s(t) = \sum_i w_i x_i(t)$ cannot align the peaks of signals that are (say) $\pi/2$ radians out of phase, even though such an alignment—prior to combining the signals—would significantly reduce the normalized prediction error in eq. (1).

A simple extension of the method overcomes this shortcoming. Given real-valued signals, $\{x_i(t)\}$, we consider the *analytic* signals, $\{\tilde{x}_i(t)\}$, whose imaginary components are computed by Hilbert transforms[10]. The Fourier series of these signals are related by:

$$x_i(t) = \sum_k \alpha_k \cos(\omega_k t + \phi_k) \quad \Longleftrightarrow \quad \tilde{x}_i(t) = \sum_k \alpha_k e^{i(\omega_k t + \phi_k)}. \tag{3}$$

We now reconsider the problem of the previous section, looking for the linear combination of analytic signals, $s(t) = \sum_i w_i \tilde{x}_i(t)$, that minimizes the cost function in eq. (1). In this setting, moreover, we allow the weights $w_i$ to be *complex* so that they can compensate for phase changes across channels. Eq. (2) generalizes in a straightforward way to:

$$\varepsilon(\boldsymbol{w}, \tau) = \frac{\sum_{ij} w_i^* w_j A_{ij}(\tau)}{\sum_{ij} w_i^* w_j B_{ij}}, \tag{4}$$

where $A(\tau)$ and $B$ are Hermitian matrices with matrix elements

$$A_{ij}(\tau) = \sum_t \Big[ \tilde{x}_i^*(t)\tilde{x}_j(t) + \tilde{x}_i^*(t+\tau)\tilde{x}_j(t+\tau) - \tilde{x}_i^*(t)\tilde{x}_j(t+\tau) - \tilde{x}_i^*(t+\tau)\tilde{x}_j(t) \Big]$$

and $B_{ij} = \sum_t \tilde{x}_i^*(t)\tilde{x}_j(t)$. Again, the optimal weights $w_i$ are given by the eigenvector corresponding to the smallest eigenvalue of the matrix $B^{-1}A(\tau)$. (Note that all the eigenvalues of this matrix are real because the matrix is Hermitian.)

Our analysis so far suggests a simple-minded approach to investigating periodic structure in speech. In particular, consider the following algorithm for pitch tracking. The first step of the algorithm is to pass speech through a cochlear filterbank and compute analytic

signals, $\tilde{x}_i(t)$, via Hilbert transforms. The next step is to diagonalize the matrices $B^{-1}A(\tau)$ on sliding windows of $\tilde{x}_i(t)$ over a range of pitch periods, $\tau \in [\tau_{\min}, \tau_{\max}]$. The final step is to estimate the pitch periods by the values of $\tau$ that minimize the cost function, eq. (1), for each sliding window. One might expect such an algorithm to be relatively robust to noise (because it can zero the weights of corrupted channels), as well as insensitive to phase changes across channels (because it can absorb them with complex weights).

Despite these attractive features, the above algorithm has serious deficiencies. Its worst shortcoming is the amount of computation needed to estimate the pitch period, $\tau$. Note that the analysis step requires computing $n^2$ cross-correlation functions, $\sum_t \tilde{x}_i^*(t)\tilde{x}_j(t+\tau)$, and diagonalizing the $n \times n$ matrix, $B^{-1}A(\tau)$. This step is unwieldy for three reasons: (i) the burden of recomputing cross-correlations for different values of $\tau$, (ii) the high sampling rates required to avoid aliasing in upper frequency bands, and (iii) the poor scaling with the number of channels, $n$. We address these concerns in the following sections.

### 2.3 Extracting the fundamental

Further signal processing is required to create multichannel output whose periodic structure can be analyzed more efficiently. Our front end, shown in Fig. 1, is designed to analyze voiced speech with fundamental frequencies in the range $f_0 \in [f_{\min}, f_{\max}]$, where $f_{\max} < 2f_{\min}$. The one-octave restriction on $f_0$ can be lifted by considering parallel, overlapping implementations of our front end for different frequency octaves.

The stages in our front end are inspired by important aspects of auditory processing[10]. Cochlear filtering is modeled by a Bark scale filterbank with contiguous passbands. Next, we compute narrowband envelopes by passing the outputs of these filters through two nonlinearities: half-wave rectification and cube-root compression. These operations are commonly used to model the compressive unidirectional response of inner hair cells to movement along the basilar membrane. Evidence for comparison of envelopes in the peripheral auditory system comes from experiments on *comodulation masking release*[11]. Thus, the next stage of our front end creates a multichannel array of signals by pairwise multiplying envelopes from nearby parts of the frequency spectrum. Allowed pairs consist of any two envelopes, including an envelope with itself, that might in principle contain energy at two consecutive harmonics of the fundamental. Multiplying these harmonics—just like multiplying two sine waves—produces intermodulation distortion with energy at the sum and difference frequencies. The energy at the difference frequency creates a signature of "residue" pitch at $f_0$. The energy at the sum frequency is removed by bandpass filtering to frequencies $[f_{\min}, f_{\max}]$ and aggressively downsampling to a sampling rate $f_s = 4f_{\min}$. Finally, we use Hilbert transforms to compute the analytic signal in each channel, which we call $\tilde{\chi}_i(t)$.

In sum, the stages of the front end create an array of bandlimited analytic signals, $\tilde{\chi}_i(t)$, that—while derived from different parts of the frequency spectrum—have energy concentrated at the fundamental frequency, $f_0$. Note that the bandlimiting of these channels to frequencies $[f_{\min}, f_{\max}]$ where $f_{\max} < 2f_{\min}$ removes the possibility that a channel contains periodic energy at any harmonic other than the fundamental. *In voiced speech, this has the effect that periodic channels contain noisy sine waves with frequency $f_0$.*

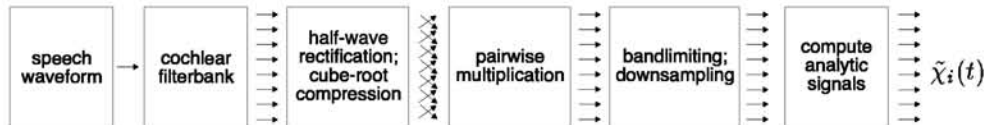

Figure 1: Signal processing in the front end.

How can we combine these "baseband" signals to enhance the periodic signature of a speaker's pitch? The nature of these signals leads to an important simplification of the problem. As opposed to measuring the autocorrelation at lag $\tau$, as in eq. (1), here we can measure the periodicity of the combined signal by a *simple sinusoidal fit*. Let $\Delta = 2\pi f_0 / f_s$ denote the phase accumulated per sample by a sine wave with frequency $f_0$ at sampling rate $f_s$, and let $s(t) = \sum_i w_i \tilde{\mathcal{X}}_i(t)$ denote the combined signal. We measure the periodicity of the combined signal by

$$\varepsilon(\boldsymbol{w}, \Delta) = \frac{\sum_t |s(t+1) - s(t)e^{i\Delta}|^2}{\sum_t |s(t)|^2} = \frac{\sum_{ij} w_i^* w_j A_{ij}(\Delta)}{\sum_{ij} w_i^* w_j B_{ij}}, \qquad (5)$$

where the matrix $B$ is again formed by computing equal-time cross-correlations, and the matrix $A(\Delta)$ has elements

$$A_{ij}(\Delta) = \sum_t \left[ \tilde{\chi}_i^*(t)\tilde{\chi}_j(t) + \tilde{\chi}_i^*(t+1)\tilde{\chi}_j(t+1) - e^{-i\Delta}\tilde{\chi}_i^*(t)\tilde{\chi}_j(t+1) - e^{i\Delta}\tilde{\chi}_i^*(t+1)\tilde{\chi}_j(t) \right].$$

For fixed $\Delta$, the optimal weights $w_i$ are given by the eigenvector corresponding to the smallest eigenvalue of the matrix $B^{-1}A(\Delta)$.

Note that optimizing the cost function in eq. (5) over the phase, $\Delta$, is equivalent to optimizing over the fundamental frequency, $f_0$, or the pitch period, $\tau$. The structure of this cost function makes it much easier to optimize than the earlier measure of periodicity in eq. (1). For instance, the matrix elements $A_{ij}(\Delta)$ depend only on the equal-time and one-sample-lagged cross-correlations, which do not need to be recomputed for different values of $\Delta$. Also, the channels $\tilde{\chi}_i(t)$ appearing in this cost function are sampled at a clock rate on the order of $f_0$, as opposed to the original sampling rate of the speech. Thus, the few cross-correlations that are required can be computed with many fewer operations. These properties lead to a more efficient algorithm than the one in the previous section. The improved algorithm, working with baseband signals, estimates the pitch by optimizing eq. (5) over $\boldsymbol{w}$ and $\Delta$ for sliding windows of $\tilde{\chi}_i(t)$. One problem still remains, however—the need to invert and diagonalize large numbers of $n \times n$ matrices, where the number of channels, $n$, may be prohibitively large. This final obstacle is removed in the next section.

### 2.4 Hierarchical analysis

We have developed a fast recursive algorithm to locate a good approximation to the minimum of eq. (5). The recursive algorithm works by constructing and diagonalizing $2 \times 2$ matrices, as opposed to the $n \times n$ matrices required for an exact solution. Our approximate algorithm also provides a hierarchical analysis of the frequency spectrum that is interesting in its own right. A sketch of the algorithm is given below.

The base step of the recursion estimates a value $\Delta_i$ for each individual channel by minimizing the error of a sinusoidal fit:

$$\varepsilon_i(\Delta_i) = \sum_t \left| \tilde{\chi}_i(t+1) - \tilde{\chi}_i(t)e^{i\Delta_i} \right|^2 \Big/ \sum_t \left| \tilde{\chi}_i(t) \right|^2 . \qquad (6)$$

The minimum of the right hand side can be computed by setting its derivative to zero and solving a quadratic equation in the variable $e^{i\Delta_i}$. If this minimum does not correspond to a legitimate value of $f_0 \in [f_{\min}, f_{\max}]$, the $i$th channel is discarded from future analysis, effectively setting its weight $w_i$ to zero. Otherwise, the algorithm passes three arguments to a higher level of the recursion: the values of $\Delta_i$ and $\varepsilon_i(\Delta_i)$, and the channel $\tilde{\chi}_i(t)$ itself.

The recursive step of the algorithm takes as input two auditory "substreams", $s_l(t)$ and $s_u(t)$, derived from "lower" and "upper" parts of the frequency spectrum, and returns as output a single combined stream, $s(t) = w_l s_l(t) + w_u s_u(t)$. In the first step

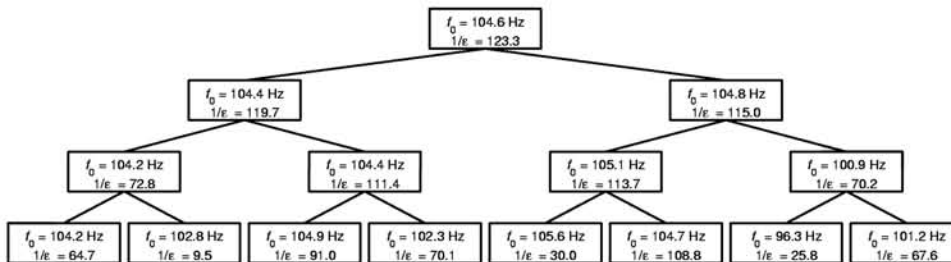

Figure 2: Measures of pitch ($f_0$) and periodicity ($\varepsilon^{-1}$) in nested regions of the frequency spectrum. The nodes in this tree describe periodic structure in the vowel /u/ from 400–1080 Hz. The nodes in the first (bottom) layer describe periodicity cues in individual channels; the nodes in the $k$th layer measure cues integrated across $2^{k-1}$ channels.

of the recursion, the substreams correspond to individual channels $\tilde{\chi}_i(t)$, while in the $k$th step, they correspond to weighted combinations of $2^{k-1}$ channels. Associated with the substreams are phases, $\Delta_l$ and $\Delta_u$, corresponding to estimates of $f_0$ from different parts of the frequency spectrum. The combined stream is formed by optimizing eq.(5) over the two-component weight vector, $w = [w_l, w_u]$. Note that the eigenvalue problem in this case involves only a $2 \times 2$ matrix, as opposed to an $n \times n$ matrix. The value of $\Delta$ determines the period of the combined stream; in practice, we optimize it over the interval defined by $\Delta_l$ and $\Delta_u$. Conveniently, this interval tends to shrink at each level of the recursion.

The algorithm works in a bottom-up fashion. Channels are combined pairwise to form streams, which are in turn combined pairwise to form new streams. Each stream has a pitch period and a measure of periodicity computed by optimizing eq. (5). We order the channels so that streams are derived from contiguous (or nearly contiguous) parts of the frequency spectrum. Fig. 2 shows partial output of this recursive procedure for a windowed segment of the vowel /u/. Note how as one ascends the tree, the combined streams have greater periodicity and less variance in their pitch estimates. This shows explicitly how the algorithm integrates information across narrow frequency bands of speech. The recursive output also suggests a useful representation for studying problems, such as speaker separation, that depend on grouping different parts of the spectrum by their estimates of $f_0$.

## 3 Experiments

We investigated the performance of our algorithm in simple experiments on synthesized vowels. Fig. 3 shows results from experiments on the vowel /u/. The pitch contours in these plots were computed by the recursive algorithm in the previous section, with $f_{\min} = 80$ Hz, $f_{\max} = 140$ Hz, and 60 ms windows shifted in 10 ms intervals. The solid curves show the estimated pitch contour for the clean wideband waveform, sampled at 8 kHz. The left panel shows results for filtered versions of the vowel, bandlimited to four different frequency octaves. These plots show that the algorithm can extract the pitch from different parts of the frequency spectrum. The right panel shows the estimated pitch contours for the vowel in 0 dB white noise and four types of -20 dB bandlimited noise. The signal-to-noise ratios were computed from the ratio of (wideband) speech energy to noise energy. The white noise at 0 dB presents the most difficulty; by contrast, the bandlimited noise leads to relatively few failures, even at -20 dB. Overall, the algorithm is quite robust to noise and filtering. (Note that the particular frequency octaves used in these experiments had no special relation to the filters in our front end.) The pitch contours could be further improved by some form of smoothing, but this was not done for the plots shown.

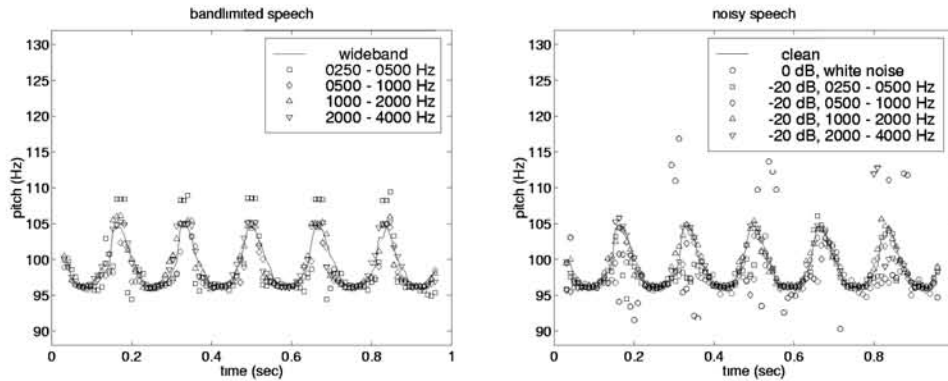

Figure 3: Tracking the pitch of the vowel /u/ in corrupted speech.

## 4 Discussion

Many aspects of this work need refinement. Perhaps the most important is the initial filtering into narrow frequency bands. While narrow filters have the ability to resolve individual harmonics, overly narrow filters—which reduce all speech input to sine waves—do not adequately differentiate periodic versus noisy excitation. We hope to replace the Bark scale filterbank in Fig. 1 by one that optimizes this tradeoff. We also want to incorporate adaptation and gain control into the front end, so as to improve the performance in nonstationary listening conditions. Finally, beyond the problem of pitch tracking, we intend to develop the hierarchical representation shown in Fig. 2 for harder problems in phoneme recognition and speaker separation[7]. These harder problems seem to require a method, like ours, that decomposes the frequency spectrum into its periodic and non-periodic components.

## References

[1] Stevens, K. N. 1999. *Acoustic Phonetics*. MIT Press: Cambridge, MA.

[2] Miller, G. A. and Nicely, P. E. 1955. An analysis of perceptual confusions among some English consonants. *Journal of the Acoustical Society of America* **27**, 338–352.

[3] Bregman, A. S. 1994. *Auditory Scene Analysis: the Perceptual Organization of Sound*. MIT Press: Cambridge, MA.

[4] Brokx, J. P. L. and Noteboom, S. G. 1982. Intonation and the perceptual separation of simultaneous voices. *J. Phonetics* **10**, 23–26.

[5] Hess, W. 1983. *Pitch Determination of Speech Signals: Algorithms and Devices*. Springer-Verlag.

[6] Talkin, D. 1995. A Robust Algorithm for Pitch Tracking (RAPT). In Kleijn, W. B. and Paliwal, K. K. (Eds.), *Speech Coding and Synthesis*, 497–518. Elsevier Science.

[7] Roweis, S. 2000. One microphone source separation. In Tresp, V., Dietterich, T., and Leen, T. (Eds.), *Advances in Neural Information Processing Systems* **13**. MIT Press: Cambridge, MA.

[8] Slaney, M. and Lyon, R. F. 1990. A perceptual pitch detector. In *Proc. ICASSP-90*, **1**, 357–360.

[9] Molgedey, L. and Schuster, H. G. 1994. Separation of a mixture of independent signals using time delayed correlations. *Phys. Rev. Lett.* **72**(23), 3634–3637.

[10] Hartmann, W. A. 1997. *Signals, Sound, and Sensation*. Springer-Verlag.

[11] Hall, J. W., Haggard, M. P., and Fernandes, M. A. 1984. Detection in noise by spectro-temporal pattern analysis. *J. Acoust. Soc. Am.* **76**, 50–56.
